# Bayesian Query Construction for Neural Network Models

**Gerhard Paass**          **Jörg Kindermann**
German National Research Center for Computer Science (GMD)
D-53757 Sankt Augustin, Germany
paass@gmd.de                    kindermann@gmd.de

## Abstract

If data collection is costly, there is much to be gained by actively selecting particularly informative data points in a sequential way. In a Bayesian decision-theoretic framework we develop a query selection criterion which explicitly takes into account the intended use of the model predictions. By Markov Chain Monte Carlo methods the necessary quantities can be approximated to a desired precision. As the number of data points grows, the model complexity is modified by a Bayesian model selection strategy. The properties of two versions of the criterion are demonstrated in numerical experiments.

## 1   INTRODUCTION

In this paper we consider the situation where data collection is costly, as when for example, real measurements or technical experiments have to be performed. In this situation the approach of query learning ('active data selection', 'sequential experimental design', etc.) has a potential benefit. Depending on the previously seen examples, a new input value ('query') is selected in a systematic way and the corresponding output is obtained. The motivation for query learning is that random examples often contain redundant information, and the concentration on non-redundant examples must necessarily improve generalization performance.

We use a Bayesian decision-theoretic framework to derive a criterion for query construction. The criterion reflects the intended use of the predictions by an appropriate

loss function. We limit our analysis to the selection of the next data point, given a set of data already sampled. The proposed procedure derives the expected loss for candidate inputs and selects a query with minimal expected loss.

There are several published surveys of query construction methods [Ford et al. 89, Plutowski White 93, Sollich 94]. Most current approaches, e.g. [Cohn 94], rely on the information matrix of parameters. Then however, all parameters receive equal attention regardless of their influence on the intended use of the model [Pronzato Walter 92]. In addition, the estimates are valid only asymptotically. Bayesian approaches have been advocated by [Berger 80], and applied to neural networks [MacKay 92]. In [Sollich Saad 95] their relation to maximum information gain is discussed. In this paper we show that by using Markov Chain Monte Carlo methods it is possible to determine all quantities necessary for the selection of a query. This approach is valid in small sample situations, and the procedure's precision can be increased with additional computational effort. With the square loss function, the criterion is reduced to a variant of the familiar integrated mean square error [Plutowski White 93].

In the next section we develop the query selection criterion from a decision-theoretic point of view. In the third section we show how the criterion can be calculated using Markov Chain Monte Carlo methods and we discuss a strategy for model selection. In the last section, the results of two experiments with MLPs are described.

## 2   A DECISION-THEORETIC FRAMEWORK

Assume we have an input vector $x$ and a scalar output $y$ distributed as $y \sim p(y \mid x, w)$ where $w$ is a vector of parameters. The conditional expected value is a deterministic function $f(x, w) := E(y \mid x, w)$ where $y = f(x, w) + \epsilon$ and $\epsilon$ is a zero mean error term. Suppose we have iteratively collected observations $D_{(n)} := ((\tilde{x}_1, \tilde{y}_1), \ldots, (\tilde{x}_n, \tilde{y}_n))$. We get the Bayesian posterior $p(w \mid D_{(n)}) = p(D_{(n)} \mid w) \, p(w) / \int p(D_{(n)} \mid w) \, p(w) \, dw$ and the predictive distribution $p(y \mid x, D_{(n)}) = \int p(y \mid x, w) \, p(w \mid D_{(n)}) \, dw$ if $p(w)$ is the prior distribution.

We consider the situation where, based on some data $x$, we have to perform an action $\alpha$ whose result depends on the unknown output $y$. Some decisions may have more severe effects than others. The loss function $L(y, \alpha) \in [0, \infty)$ measures the loss if $y$ is the true value and we have taken the action $\alpha \in \mathcal{A}$. In this paper we consider real-valued actions, e.g. setting the temperature $\alpha$ in a chemical process. We have to select an $\alpha \in \mathcal{A}$ only knowing the input $x$. According to the *Bayes Principle* [Berger 80, p.14] we should follow a decision rule $d : x \to \alpha$ such that the average risk $\int R(w, d) \, p(w \mid D_{(n)}) \, dw$ is minimal, where the risk is defined as $R(w, d) := \int L(y, d(x)) \, p(y \mid x, w) \, p(x) \, dy \, dx$. Here $p(x)$ is the distribution of future inputs, which is assumed to be known.

For the *square loss* function $L(y, \alpha) = (y - \alpha)^2$, the conditional expectation $d(x) := E(y \mid x, D_{(n)})$ is the optimal decision rule. In a control problem the loss may be larger at specific critical points. This can be addressed with a *weighted square loss* function $L(y, \alpha) := h(y)(y - \alpha)^2$, where $h(y) \geq 0$ [Berger 80, p.111]. The expected loss for an action is $\int (y - \alpha)^2 h(y) \, p(y \mid x, D_{(n)}) \, dy$. Replacing the predictive density $p(y \mid x, D_{(n)})$ with the weighted predictive density

$\tilde{p}(y \mid x, D_{(n)}) := h(y) \, p(y \mid x, D_{(n)})/G(x)$, where $G(x) := \int h(y) \, p(y \mid x, D_{(n)}) \, dy$, we get the optimal decision rule $d(x) := \int y \tilde{p}(y \mid x, D_{(n)}) \, dy$ and the average loss $G(x) \int (y - E(y \mid x, D_{(n)}))^2 \, \tilde{p}(y \mid x, D_{(n)}) \, dy$ for a given input $x$. With these modifications, all later derivations for the square loss function may be applied to the weighted square loss.

The aim of query sampling is the selection of a new observation $\breve{x}$ in such a way that the average risk will be maximally reduced. Together with its still unknown $y$-value, $\breve{x}$ defines a new observation $(\breve{x}, \breve{y})$ and new data $D_{(n)} \cup (\breve{x}, \breve{y})$. To determine this risk for some given $\breve{x}$ we have to perform the following conceptual steps for a candidate query $\breve{x}$:

1. *Future Data:* Construct the possible sets of 'future' observations $D_{(n)} \cup (\breve{x}, \breve{y})$, where $\breve{y} \sim p(y \mid \breve{x}, D_{(n)})$.

2. *Future posterior:* Determine a 'future' posterior distribution of parameters $p(w \mid D_{(n)} \cup (\breve{x}, \breve{y}))$ that depends on $\breve{y}$ in the same way as though it had actually been observed.

3. *Future Loss:* Assuming $d^*_{\breve{y}, \breve{x}}(x)$ is the optimal decision rule for given values of $\breve{x}$, $\breve{y}$, and $x$, compute the resulting loss as

$$\bar{r}^*_{\breve{y}, \breve{x}}(x) := \int L(y, d^*_{\breve{y}, \breve{x}}(x)) \, p(y \mid x, w) \, p(w \mid D_{(n)} \cup (\breve{x}, \breve{y})) \, dy \, dw \qquad (1)$$

4. *Averaging:* Integrate this quantity over the future trial inputs $x$ distributed as $p(x)$ and the different possible future outputs $\breve{y}$, yielding
$\bar{r}^*_{\breve{x}} := \int \bar{r}^*_{\breve{y}, \breve{x}}(x) \, p(x) \, p(\breve{y} \mid \breve{x}, D_{(n)}) \, dx \, d\breve{y}$.

This procedure is repeated until an $\breve{x}$ with minimal average risk is found. Since local optima are typical, a global optimization method is required. Subsequently we then try to determine whether the current model is still adequate or whether we have to increase its complexity (e.g. by adding more hidden units).

# 3 COMPUTATIONAL PROCEDURE

Let us assume that the real data $D_{(n)}$ was generated according to a regression model $y = f(x, w) + \epsilon$ with i.i.d. Gaussian noise $\epsilon \sim N(0, \sigma^2(w))$. For example $f(x, w)$ may be a multilayer perceptron or a radial basis function network. Since the error terms are independent, the posterior density is $p(w \mid D_{(n)}) \propto p(w) \prod_{i=1}^{n} p(\tilde{y}_i \mid \tilde{x}_i, w)$ even in the case of query sampling [Ford et al. 89].

As the analytic derivation of the posterior is infeasible except in trivial cases, we have to use approximations. One approach is to employ a normal approximation [MacKay 92], but this is unreliable if the number of observations is small compared to the number of parameters. We use Markov Chain Monte Carlo procedures [Paaß 91, Neal 93] to generate a sample $W_{(B)} := \{w_1, \ldots w_B\}$ of parameters distributed according to $p(w \mid D_{(n)})$. If the number of sampling steps approaches infinity, the distribution of the simulated $w_b$ approximates the posterior arbitrarily well.

To take into account the range of future $\breve{y}$-values, we create a set of them by simulation. For each $w_b \in W_{(B)}$ a number of $\breve{y} \sim p(y \mid \breve{x}, w_b)$ is generated. Let

$\check{Y}_{(\check{x},R)} := \{\check{y}_1, \ldots, \check{y}_R\}$ be the resulting set. Instead of performing a new Markov Monte Carlo run to generate a new sample according to $p(w \mid D_{(n)} \cup (\check{x}, \check{y}))$, we use the old set $W_{(B)}$ of parameters and reweight them (importance sampling). In this way we may approximate integrals of some function $g(w)$ with respect to $p(w \mid D_{(n)} \cup (\check{x}, \check{y}))$ [Kalos Whitlock 86, p.92]:

$$\int g(w) \, p(w \mid D_{(n)} \cup (\check{x}, \check{y})) \, dw \quad \approx \quad \frac{\sum_{b=1}^{B} g(w_b) \, p(\check{y} \mid \check{x}, w_b)}{\sum_{b=1}^{B} p(\check{y} \mid \check{x}, w_b)} \tag{2}$$

The approximation error approaches zero as the size of $W_{(B)}$ increases.

## 3.1 APPROXIMATION OF FUTURE LOSS

Consider the future loss $\bar{r}^*_{\check{y},\check{x}}(x)$ given new observation $(\check{x}, \check{y})$ and trial input $x_t$. In the case of the square loss function, (1) can be transformed to

$$\bar{r}^*_{\check{y},\check{x}}(x_t) \quad = \quad \int [f(x_t, w) - E(y \mid x_t, D_{(n)} \cup (\check{x}, \check{y}))]^2 \, p(w \mid D_{(n)} \cup (\check{x}, \check{y})) \, dw \tag{3}$$

$$+ \int \sigma^2(w) \, p(w \mid D_{(n)} \cup (\check{x}, \check{y})) \, dw$$

where $\sigma^2(w) := \mathrm{Var}(y \mid x, w)$ is independent of $x$. Assume a set $X_T = \{x_1, \ldots, x_T\}$ is given, which is representative of trial inputs for the distribution $p(x)$. Define $S(\check{x}, \check{y}) := \sum_{b=1}^{B} p(\check{y} \mid \check{x}, w_b)$ for $\check{y} \in \check{Y}_{(\check{x},R)}$. Then from equations (2) and (3) we get $\hat{E}(y \mid x_t, D_{(n)} \cup (\check{x}, \check{y})) := 1/S(\check{x}, \check{y}) \sum_{b=1}^{B} f(x_t, w_b) \, p(\check{y} \mid \check{x}, w_b)$ and

$$\bar{r}^*_{\check{y},\check{x}}(x_t) \quad \approx \quad \frac{1}{S(\check{x}, \check{y})} \sum_{b=1}^{B} \sigma^2(w_b) \, p(\check{y} \mid \check{x}, w_b) \tag{4}$$

$$+ \frac{1}{S(\check{x}, \check{y})} \sum_{b=1}^{B} [f(x_t, w_b) - \hat{E}(y \mid x_t, D_{(n)} \cup (\check{x}, \check{y}))]^2 \, p(\check{y} \mid \check{x}, w_b)$$

The final value of $\bar{r}^*_{\check{x}}$ is obtained by averaging over the different $\check{y} \in \check{Y}_{(\check{x},R)}$ and different trial inputs $x_t \in X_T$. To reduce the variance, the trial inputs $x_t$ should be selected by importance sampling (2) to concentrate them on regions with high current loss (see (5) below). To facilitate the search for an $\check{x}$ with minimal $\bar{r}^*_{\check{x}}$ we reduce the extent of random fluctuations of the $\check{y}$ values. Let $(v_1, \ldots, v_R)$ be a vector of random numbers $v_r \sim N(0,1)$, and let $j_r$ be randomly selected from $\{1, \ldots, B\}$. Then for each $\check{x}$ the possible observations $\check{y}_r \in \check{Y}_{(\check{x},R)}$ are defined as $\check{y}_r := f(\check{x}, w_{j_r}) + v_r \sigma^2(w_{j_r})$. In this way the difference between neighboring inputs is not affected by noise, and search procedures can exploit gradients.

## 3.2 CURRENT LOSS

As a proxy for the future loss, we may use the current loss at $\check{x}$,

$$r_{curr}(\check{x}) = p(\check{x}) \int L(y, d^*(\check{x})) \, p(y \mid \check{x}, D_{(n)}) \, dy \tag{5}$$

where $p(\breve{x})$ weights the inputs according to their relevance. For the square loss function the average loss at $\breve{x}$ is the conditional variance $\text{Var}(y \mid \breve{x}, D_{(n)})$. We get

$$r_{curr}(\breve{x}) \;=\; p(\breve{x}) \int (f(\breve{x}, w) - E(y \mid \breve{x}, D_{(n)}))^2 \, p(w \mid D_{(n)}) \, dw \qquad (6)$$

$$+ \, p(\breve{x}) \int \sigma^2(w) \, p(w \mid D_{(n)}) \, dw$$

If $\hat{E}(y \mid \breve{x}, D_{(n)}) := \frac{1}{B} \sum_{b=1}^{B} f(\breve{x}, w_b)$ and the sample $W_{(B)} := \{w_1, \ldots, w_B\}$ is representative of $p(w \mid D_{(n)})$ we can approximate the current loss with

$$r_{curr}(\breve{x}) \;\approx\; \frac{p(\breve{x})}{B} \sum_{b=1}^{B} (f(\breve{x}, w_b) - \hat{E}(y \mid \breve{x}, D_{(n)}))^2 + \frac{p(\breve{x})}{B} \sum_{b=1}^{B} \sigma^2(w_b) \qquad (7)$$

If the input distribution $p(x)$ is uniform, the second term is independent of $\breve{x}$.

## 3.3 COMPLEXITY REGULARIZATION

Neural network models can represent arbitrary mappings between finite-dimensional spaces if the number of hidden units is sufficiently large [Hornik Stinchcombe 89]. As the number of observations grows, more and more hidden units are necessary to catch the details of the mapping. Therefore we use a sequential procedure to increase the capacity of our networks during query learning. White and Wooldridge call this approach the "method of sieves" and provide some asymptotic results on its consistency [White Wooldridge 91]. Gelfand and Dey compare Bayesian approaches for model selection and prove that, in the case of nested models $M_1$ and $M_2$, model choice by the ratio of popular Bayes factors $p(D_{(n)} \mid M_i) := \int p(D_{(n)} \mid w, M_i) p(w \mid M_i) \, dw$ will *always* choose the full model regardless of the data as $n \to \infty$ [Gelfand Dey 94]. They show that the *pseudo-Bayes factor*, a Bayesian variant of crossvalidation, is not affected by this paradox

$$\lambda(M_1, M_2) := \prod_{j=1}^{n} p(\tilde{y}_j \mid \tilde{x}_j, D_{(n,j)}, M_1) / \prod_{j=1}^{n} p(\tilde{y}_j \mid \tilde{x}_j, D_{(n,j)}, M_2) \qquad (8)$$

Here $D_{(n,j)} := D_{(n)} \backslash (\tilde{x}_j, \tilde{y}_j)$. As the difference between $p(w \mid D_{(n)})$ and $p(w \mid D_{(n,j)})$ is usually small, we use the full posterior as the importance function (2) and get

$$p(\tilde{y}_j \mid \tilde{x}_j, D_{(n,j)}, M_i) \;=\; \int p(\tilde{y}_j \mid \tilde{x}_j, w, M_i) \, p(w \mid D_{(n,j)}, M_i) \, dw$$

$$\approx \; B / \left( \sum_{b=1}^{B} 1 / p(\tilde{y}_j \mid \tilde{x}_j, w_b, M_i) \right) \qquad (9)$$

## 4  NUMERICAL DEMONSTRATION

In a first experiment we tested the approach for a small a 1-2-1 MLP target function with Gaussian noise $N(0, 0.05^2)$. We assumed the square loss function and a uniform input distribution $p(x)$ over $[-5, 5]$. Using the "true" architecture for the approximating model we started with a single randomly generated observation. We

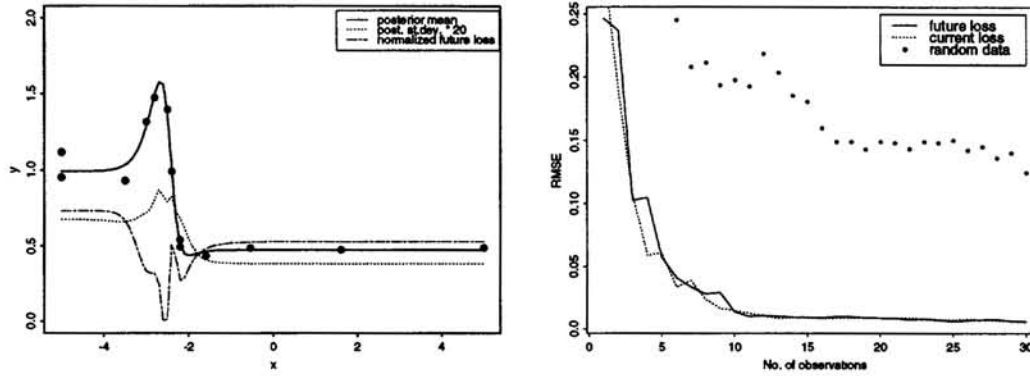

Figure 1: Future loss exploration: predicted posterior mean, future loss and current loss for 12 observations (left), and root mean square error of prediction (right).

estimated the future loss by (4) for 100 different inputs and selected the input with smallest future loss as the next query. $B = 50$ parameter vectors were generated requiring 200,000 Metropolis steps. Simultaneously we approximated the current loss criterion by (7). The left side of figure 1 shows the typical relation of both measures. In most situations the future loss is low in the same regions where the current loss (posterior standard deviation of mean prediction) is high. The queries are concentrated in areas of high variation and the estimated posterior mean approximates the target function quite well.

In the right part of figure 1 the RMSE of prediction averaged over 12 independent experiments is shown. After a few observations the RMSE drops sharply. In our example there is no marked difference between the prediction errors resulting from the future loss and the current loss criterion (also averaged over 12 experiments). Considering the substantial computing effort this favors the current loss criterion. The dots indicate the RMSE for randomly generated data (averaged over 8 experiments) using the same Bayesian prediction procedure. Because only few data points were located in the critical region of high variation the RMSE is much larger.

In the second experiment, a 2-3-1 MLP defined the target function $f(x, w_0)$, to which Gaussian noise of standard deviation 0.05 was added. $f(x, w_0)$ is shown in the left part of figure 2. We used five MLPs with 2-6 hidden units as candidate models $M_1, \ldots, M_5$ and generated $B = 45$ samples $W_{(B)}$ of the posterior $p(w \mid D_{(n)}, M_i)$, where $D_{(n)}$ is the current data. We started with 30,000 Metropolis steps for small values of $n$ and increased this to 90,000 Metropolis steps for larger values of $n$. For a network with 6 hidden units and $n = 50$ observations, 10,000 Metropolis steps took about 30 seconds on a Sparc10 workstation. Next, we used equation (9) to compare the different models, and then used the optimal model to calculate the current loss (7) on a regular grid of $41 \times 41 = 1681$ query points $\check{x}$. Here we assumed the square loss function and a uniform input distribution $p(x)$ over $[-5, 5] \times [-5, 5]$. We selected the query point with maximal current loss and determined the final query point with a hillclimbing algorithm. In this way we were rather sure to get close to the true global optimum.

The main result of the experiment is summarized in the right part of figure 2. It

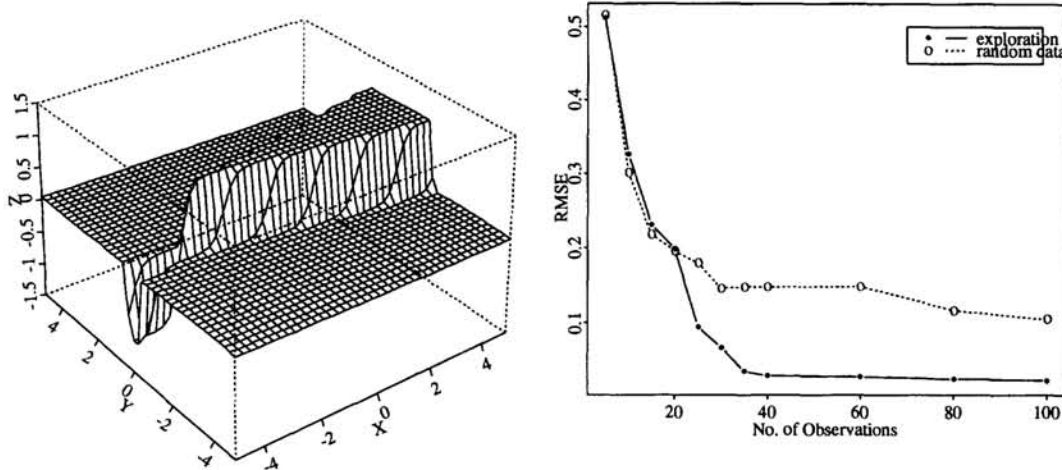

Figure 2: Current loss exploration: MLP target function and root mean square error.

shows - averaged over 3 experiments - the root mean square error between the true mean value and the posterior mean $E(y \mid x)$ on the grid of 1681 inputs in relation to the sample size. Three phases of the exploration can be distinguished (see figure 3). In the beginning a search is performed with many queries on the border of the input area. After about 20 observations the algorithm knows enough detail about the true function to concentrate on the relevant parts of the input space. This leads to a marked reduction of the mean square error. After 40 observations the systematic part of the true function has been captured nearly perfectly. In the last phase of the experiment the algorithm merely reduces the uncertainty caused by the random noise. In contrast, the data generated randomly does not have sufficient information on the details of $f(x, w)$, and therefore the error only gradually decreases. Because of space constraints we cannot report experiments with radial basis functions which led to similar results.

## Acknowledgements

This work is part of the joint project 'REFLEX' of the German Fed. Department of Science and Technology (BMFT), grant number 01 IN 111A/4. We would like to thank Alexander Linden, Mark Ring, and Frank Weber for many fruitful discussions.

## References

[Berger 80] Berger, J. (1980): *Statistical Decision Theory, Foundations, Concepts, and Methods.* Springer Verlag, New York.

[Cohn 94] Cohn, D. (1994): Neural Network Exploration Using Optimal Experimental Design. In J. Cowan et al. (eds.): *NIPS 5.* Morgan Kaufmann, San Mateo.

[Ford et al. 89] Ford, I., Titterington, D.M., Kitsos, C.P. (1989): Recent Advances in Nonlinear Design. *Technometrics,* **31**, p.49-60.

[Gelfand Dey 94] Gelfand, A.E., Dey, D.K. (1994): Bayesian Model Choice: Asymptotics and Exact Calculations. *J. Royal Statistical Society B,* **56**, pp.501-514.

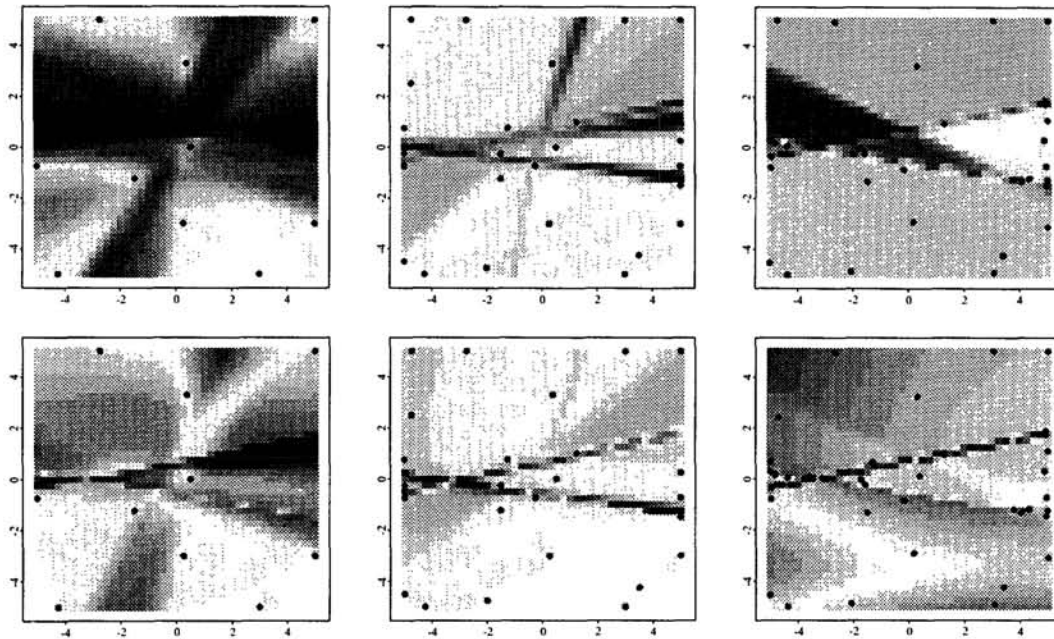

Figure 3: Squareroot of current loss (upper row) and absolute deviation from true function (lower row) for 10, 25, and 40 observations (which are indicated by dots).

[Hornik Stinchcombe 89] Hornik, K., Stinchcombe, M. (1989): Multilayer Feedforward Networks are Universal Approximators. *Neural Networks* **2**, p.359-366.

[Kalos Whitlock 86] Kalos, M.H., Whitlock, P.A. (1986): *Monte Carlo Methods,* Wiley, New York.

[MacKay 92] MacKay, D. (1992): Information-Based Objective Functions for Active Data Selection. *Neural Computation* **4**, p.590-604.

[Neal 93] Neal, R.M. (1993): *Probabilistic Inference using Markov Chain Monte Carlo Methods.* Tech. Report CRG-TR-93-1, Dep. of Computer Science, Univ. of Toronto.

[Paaß 91] Paaß, G. (1991): Second Order Probabilities for Uncertain and Conflicting Evidence. In: P.P. Bonissone et al. (eds.) *Uncertainty in Artificial Intelligence 6.* Elsevier, Amsterdam, pp. 447-456.

[Plutowski White 93] Plutowski, M., White, H. (1993): Selecting Concise Training Sets from Clean Data. *IEEE Tr. on Neural Networks,* **4**, p.305-318.

[Pronzato Walter 92] Pronzato, L., Walter, E. (1992): Nonsequential Bayesian Experimental Design for Response Optimization. In V. Fedorov, W.G. Müller, I.N. Vuchkov (eds.): *Model Oriented Data-Analysis.* Physica Verlag, Heidelberg, p. 89-102.

[Sollich 94] Sollich, P. (1994): Query Construction, Entropy and Generalization in Neural Network Models. To appear in Physical Review E.

[Sollich Saad 95] Sollich, P., Saad, D. (1995): Learning from Queries for Maximum Information Gain in Unlearnable Problems. This volume.

[White Wooldridge 91] White, H., Wooldridge, J. (1991): Some Results for Sieve Estimation with Dependent Observations. In W. Barnett et al. (eds.) : *Nonparametric and Semiparametric Methods in Econometrics and Statistics,* New York, Cambridge Univ. Press.
